# A Mathematical Programming Approach to the Kernel Fisher Algorithm

**Sebastian Mika\*, Gunnar Rätsch\*, and Klaus-Robert Müller\*+**
\*GMD FIRST.IDA, Kekuléstraße 7, 12489 Berlin, Germany
+University of Potsdam, Am Neuen Palais 10, 14469 Potsdam
{*mika, raetsch, klaus*}*@first.gmd.de*

## Abstract

We investigate a new kernel–based classifier: the Kernel Fisher Discriminant (KFD). A mathematical programming formulation based on the observation that KFD maximizes the *average margin* permits an interesting modification of the original KFD algorithm yielding the sparse KFD. We find that both, KFD and the proposed sparse KFD, can be understood in an unifying probabilistic context. Furthermore, we show connections to Support Vector Machines and Relevance Vector Machines. From this understanding, we are able to outline an interesting kernel–regression technique based upon the KFD algorithm. Simulations support the usefulness of our approach.

## 1 Introduction

Recent years have shown an enormous interest in kernel-based classification algorithms, primarily in Support Vector Machines (SVM) [2]. The success of SVMs seems to be triggered by (i) their good generalization performance, (ii) the existence of a unique solution, and (iii) the strong theoretical background: structural risk minimization [12], supporting the good empirical results. One of the key ingredients responsible for this success is the use of Mercer kernels, allowing for nonlinear decision surfaces which even might incorporate some prior knowledge about the problem to solve. For our purpose, a Mercer kernel can be defined as a function $k : \mathbb{R}^n \times \mathbb{R}^n \to \mathbb{R}$, for which some (nonlinear) mapping $\Phi : \mathbb{R}^n \to \mathcal{F}$ into a *feature space* $\mathcal{F}$ exists, such that $k(\boldsymbol{x}, \boldsymbol{y}) = (\Phi(\boldsymbol{x}) \cdot \Phi(\boldsymbol{y}))$. Clearly, the use of such kernel functions is not limited to SVMs. The interpretation as a dot–product in another space makes it particularly easy to develop new algorithms: take any (usually) linear method and reformulate it using training samples only in dot–products, which are then replaced by the kernel. Examples thereof, among others, are Kernel–PCA [9] and the Kernel Fisher Discriminant (KFD [4]; see also [8, 1]).

In this article we consider algorithmic ideas for KFD. Interestingly KFD – although exhibiting a similarly good performance as SVMs – has no explicit concept of a margin. This is noteworthy since the margin is often regarded as explanation for good generalization in SVMs. We will give an alternative formulation of KFD which makes the difference between both techniques explicit and allows a better understanding of the algorithms. Another advantage of the new formulation is that we can derive more efficient algorithms for optimizing KFDs, that have e.g. sparseness properties or can be used for regression.

## 2    A Review of Kernel Fisher Discriminant

The idea of the KFD is to solve the problem of Fisher's linear discriminant in a kernel feature space $\mathcal{F}$, thereby yielding a nonlinear discriminant in the input space. First we fix some notation. Let $\{x_i | i = 1, \dots, \ell\}$ be our training sample and $y \in \{-1, 1\}^\ell$ be the vector of corresponding labels. Furthermore define $\mathbf{1} \in \mathbb{R}^\ell$ as the vector of all ones, $\mathbf{1}_1, \mathbf{1}_2 \in \mathbb{R}^\ell$ as binary $(0, 1)$ vectors corresponding to the class labels and let $\mathcal{I}, \mathcal{I}_1$, and $\mathcal{I}_2$ be appropriate index sets over $\ell$ and the two classes, respectively (with $\ell_i = |I_i|$).

In the linear case, Fisher's discriminant is computed by maximizing the coefficient $J(w) = (w^\top S_B w)/(w^\top S_W w)$ of between and within class variance, i.e. $S_B = (m_2 - m_1)(m_2 - m_1)^\top$ and $S_W = \sum_{k=1,2} \sum_{i \in \mathcal{I}_k} (x_i - m_k)(x_i - m_k)^\top$, where $m_k$ denotes the sample mean for class $k$. To solve the problem in a kernel feature space $\mathcal{F}$ one needs a formulation which makes use of the training samples only in terms of dot–products. One first shows [4], that there exists an expansion for $w \in \mathcal{F}$ in terms of mapped training patterns, i.e.

$$w = \sum_{\mathcal{I}} \alpha_i \Phi(x_i). \tag{1}$$

Using some straight forward algebra, the optimization problem for the KFD can then be written as [5]:

$$J(\alpha) = \frac{(\alpha^\top \mu)^2}{\alpha^\top N \alpha} = \frac{\alpha^\top M \alpha}{\alpha^\top N \alpha}, \tag{2}$$

where $\mu_i = \frac{1}{\ell_i} K \mathbf{1}_i$, $N = KK^\top - \sum_{i=1,2} \ell_i \mu_i \mu_i^\top$, $\mu = \mu_2 - \mu_1$, $M = \mu \mu^\top$, and $K_{ij} = (\Phi(x_i) \cdot \Phi(x_j)) = k(x_i, x_j)$. The projection of a test point onto the discriminant is computed by $(w \cdot \Phi(x)) = \sum_{\mathcal{I}} \alpha_i k(x_i, x)$. As the dimension of the feature space is usually much higher than the number of training samples $\ell$ some form of regularization is necessary. In [4] it was proposed to add e.g. the identity or the kernel matrix $K$ to $N$, penalizing $\|\alpha\|^2$ or $\|w\|^2$, respectively (see also [3]).

There are several equivalent ways to optimize (2). One could either solve the generalized eigenproblem $M\alpha = \lambda N \alpha$, selecting the eigenvector $\alpha$ with maximal eigenvalue $\lambda$, or compute $\alpha \equiv N^{-1}(\mu_2 - \mu_1)$. Another way which will be detailed in the following exploits the special structure of problem (2).

## 3    Casting KFD into a Quadratic Program

Although there exist many efficient off-the-shelve eigensolvers or Cholesky packages which could be used to optimize (2) there remain two problems: for a large sample size $\ell$ the matrices $N$ and $M$ become unpleasantly large and the solutions $\alpha$ are non-sparse (with no obvious way to introduce sparsity in e.g. the matrix inverse). In the following we show how KFD can be cast as a convex quadratic programming problem. This new formulation will prove helpful in solving the problems mentioned above and makes it much easier to gain a deeper understanding of KFD.

As a first step we exploit the facts that the matrix $M$ is only rank one, i.e. $\alpha^\top M \alpha = (\alpha^\top (\mu_2 - \mu_1))^2$ and that with $\alpha$ any multiple of $\alpha$ is an optimal solution to (2). Thus we may fix $\alpha^\top (\mu_2 - \mu_1)$ to any non-zero value, say 2 and minimize $\alpha^\top N \alpha$. This amounts to the following quadratic program:

$$\min_{\alpha} \quad \alpha^\top N \alpha + C\,P(\alpha) \tag{3}$$

subject to:
$$\alpha^\top (\mu_2 - \mu_1) = 2. \tag{3a}$$

The regularization formerly incorporated in $N$ is made visible explicitly here through the operator P, where $C$ is a regularization constant. This program still makes use of the rather un–intuitive matrix $N$. This can be avoided by our final reformulation which can be understood as follows: Fisher's Discriminant tries to minimize the variance of the data along the projection whilst maximizing the distance between the average outputs for each

class. Considering the argumentation leading to (3) the following quadratic program does exactly this:

$$\min_{\alpha,b,\xi} \quad \|\xi\|^2 + C\,\mathrm{P}(\alpha) \tag{4}$$

subject to:
$$K\alpha + 1b = y + \xi \tag{4a}$$
$$1_i^\top \xi = 0 \text{ for } i = 1,2 \tag{4b}$$

for $\alpha,\xi \in \mathbb{R}^\ell$, and $b, C \in \mathbb{R}$, $C \geq 0$. The constraint (4a), which can be read as $(w \cdot x_i) + b = y_i + \xi_i$ for all $i \in \mathcal{I}$, pulls the output for each sample to its class–label. The term $\|\xi\|^2$ minimizes the variance of the error committed while the constraints (4b) ensure that the average output for each class is the label, i.e. for $\pm 1$ labels the average distance of the projections is two. The following proposition establishes the link to KFD:

**Proposition 1.** *For given $C \in \mathbb{R}$, any optimal solution $\alpha$ to the optimization problem* (3) *is also optimal for* (4) *and vice versa.*

The formal, rather straightforward but lengthy, proof of Proposition 1 is omitted here. It shows (i) that the feasible sets of (3) and (4) are identical with respect to $\alpha$ and (ii) that the objective functions coincide. Formulation (4) has a number of appealing properties which we will exploit in the following.

## 4   A Probabilistic Interpretation

We would like to point out the following connection (which is not specific to the formulation (4) of KFD): The Fisher discriminant is the Bayes optimal classifier for two normal distributions with equal covariance (i.e. KFD is Bayes optimal for two Gaussian in feature space.). To see this connection to Gaussians consider a regression onto the labels of the form $(w \cdot \Phi(x)) + b$, where $w$ is given by (1). Assuming a Gaussian noise model with variance $\sigma$ the likelihood can be written as

$$p(y|\alpha,\sigma^2) \equiv \exp(-\frac{1}{2\sigma^2}\sum_i ((w\cdot\Phi(x_i))+b-y_i)^2) = \exp(-\frac{1}{2\sigma^2}\|\xi\|^2).$$

Now, assume some prior $p(\alpha|C)$ over the weights with hyper-parameters $C$. Computing the posterior we would end up with the Relevance Vector Machine (RVM) [11]. An advantage of the RVM approach is that all hyper-parameters $\sigma$ and $C$ are estimated automatically. The drawback however is that one has to solve a hard, computationally expensive optimization problem. The following simplifications show how KFD can be seen as an approximation to this probabilistic approach. Assuming the noise variance $\sigma$ is known (i.e. dropping all terms depending solely on $\sigma$) and taking the logarithm of the posterior $p(y|\alpha,\sigma^2)p(\alpha|C)$, yields the following optimization problem

$$\min_{\alpha,b} \|\xi\|^2 - \log(p(\alpha|C)), \tag{5}$$

subject to the constraint (4a). Interpreting the prior as a regularization operator P, introducing an appropriate weighting factor $C$, and adding the two zero–mean constraints (4b) yields the KFD problem (4). The latter are necessary for classification as the two classes are independently assumed to be zero–mean Gaussians. This probabilistic interpretation has some appealing properties which we outline in the following:

**Interpretation of outputs**   The probabilistic framework reflects the fact, that the outputs produced by KFD can be interpreted as probabilities, thus making it possible to assign a confidence to the final classification. This is in contrast to SVMs whose outputs can not directly be seen as probabilities.

**Noise models**  In the above illustration we assumed a Gaussian noise model and some yet unspecified prior which was then interpreted as regularizer. Of course, one is not limited to Gaussian models. E.g. assuming a Laplacian noise model we would get $\|\boldsymbol{\xi}\|_1$ instead of $\|\boldsymbol{\xi}\|_2^2$ in the objective (5) or (4), respectively. Table 1 gives a selection of different noise models and their corresponding loss functions which could be used (cf. Figure 1 for an illustration). All of them still lead to convex linear or quadratic programming problems in the KFD framework.

Table 1: Loss functions for the slack variables $\boldsymbol{\xi}$ and their corresponding density/noise models in a probabilistic framework [10].

|  | loss function | density model |
|---|---|---|
| $\varepsilon$-ins. | $\|\xi\|_\varepsilon$ | $\frac{1}{2(1+\varepsilon)}\exp(-\|\xi\|_\varepsilon)$ |
| Laplacian | $\|\xi\|$ | $\frac{1}{2}\exp(-\|\xi\|)$ |
| Gaussian | $\frac{1}{2}\xi^2$ | $\frac{1}{\sqrt{2\pi}}\exp(-\frac{\xi^2}{2})$ |
| Huber's | $\begin{cases}\frac{1}{2\sigma}\xi^2 \\ \|\xi\|-\frac{\sigma}{2}\end{cases}$ | $\begin{cases}\exp(-\frac{\xi^2}{2\sigma}) & \text{if } \|\xi\| \le \sigma \\ \exp(\frac{\sigma}{2}-\|\xi\|) & \text{otherwise}\end{cases}$ |

**Regularizers**  Still open in this probabilistic interpretation is the choice of the prior or regularizer $p(\boldsymbol{\alpha}|\boldsymbol{C})$. One choice would be a zero–mean Gaussian as for the RVM. Assuming again that this Gaussians' variance $\boldsymbol{C}$ is known and a multiple of the identity this would lead to a regularizer of the form $P(\boldsymbol{\alpha}) = \|\boldsymbol{\alpha}\|^2$. Crucially, choosing a single, fixed variance parameter for all $\boldsymbol{\alpha}$ we would not achieve sparsity as in RVM anymore. But of course any other choice, e.g. from Table 1 is possible. Especially interesting is the choice of a Laplacian prior which in the optimization procedure would correspond to a $l_1$–loss on the $\boldsymbol{\alpha}$'s, i.e. $P(\boldsymbol{\alpha}) = \|\boldsymbol{\alpha}\|_1$. This choice leads to sparse solutions in the KFD as the $l_1$–norm can be seen as an approximation to the $l_0$–norm. In the following we call this particular setting *sparse KFD* (SKFD).

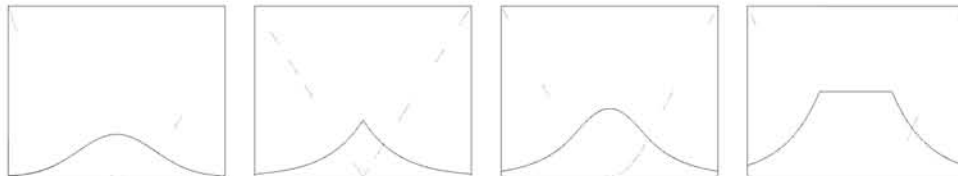

Figure 1: Illustration of Gaussian, Laplacian, Huber's robust and $\varepsilon$–insensitive loss functions (dotted) and corresponding densities (solid).

**Regression and connection to SVM**  Considering the program (4) it is rather simple to modify the KFD approach for regression. Instead of $\pm1$ outputs $\boldsymbol{y}$ we now have real–valued $\boldsymbol{y}$'s. And instead of two classes there is only one class left. Thus, we can use KFD for regression as well by simply dropping the distinction between classes in constraint (4b). The remaining constraint requires the average error to be zero while the variance of the errors is minimized.

This as well gives a connection to SVM regression (e.g. [12]), where one uses the $\varepsilon$–insensitive loss for $\boldsymbol{\xi}$ (cf. Table 1) and a $K$–regularizer, i.e. $P(\boldsymbol{\alpha}) = \boldsymbol{\alpha}^\top K \boldsymbol{\alpha} = \|\boldsymbol{w}\|^2$. Finally, we can as well draw the connection to a SVM classifier. In SVM classification one is maximizing the (smallest) margin, traded off against the complexity controlled by $\|\boldsymbol{w}\|^2$. Contrary, besides parallels in the algorithmic formulation, in KFD is no explicit concept of a margin. Instead, implicitly the *average* margin, i.e. the average distance of samples from different classes, is maximized.

**Optimization**  Besides a more intuitive understanding, the formulation (4) allows for deriving more efficient algorithms as well. Using a sparsity regularizer (i.e. SKFD) one could

employ chunking techniques during the optimization of (4). However, the problem of selecting a good working set is not solved yet, and contrary to e.g. SVM, for KFD all samples will influence the final solution via the constraints (4a), not just the ones with $\alpha_i \neq 0$. Thus these samples can not simply be eliminated from the optimization problem. Another interesting option induced by (4) is to use a sparsity regularizer *and* a linear loss function, e.g. the Laplacian loss (cf. Table 1). This results in a linear program which we call linear sparse KFD (LSKFD). This can very efficiently be solved by column generation techniques known from mathematical programming. A final possibility to optimize (4) for the standard KFD problem (i.e. quadratic loss and regularizer) is described in [6]. Here one uses a greedy approximation scheme which iteratively constructs a (sparse) solution to the full problem. Such an approach is straight forward to implement and much faster than solving a quadratic program, provided that the number of non–zero $\alpha$'s necessary to get a good approximation to the full solution is small.

## 5 Experiments

In this section we present some experimental results targeting at (i) showing that the KFD and some of its variants proposed here are capable of producing state of the art results and (ii) comparing the influence of different settings for the regularization $P(\alpha)$ and the loss–function applied to $\xi$ in kernel based classifiers.

**The Output Distribution**   In an initial experiment we compare the output distributions generated by a SVM and the KFD (cf. Figure 2). By maximizing the smallest margin and using linear slack variables for patterns which do not achieve a reasonable margin, the SVM produces a training output sharply peaked around $\pm 1$ with Laplacian tails inside the margin area (the *inside* margin area is the interval $[-1, 1]$, the *outside* area its complement). Contrary, KFD produces normal distributions which have a small variance along the discriminating direction. Comparing the distributions on the training set to those on the test set, there is almost no difference for KFD. In this sense the direction found on the training data is consistent with the test data. For SVM the output distribution on the test set is significantly different. In the example given in Figure 2 the KFD performed slightly better than SVM (1.5% vs. 1.7%; for both the best parameters found by 5-fold cross validation were used), a fact that is surprising looking only on the training distribution (which is perfectly separated for SVM but has some overlap for KFD).

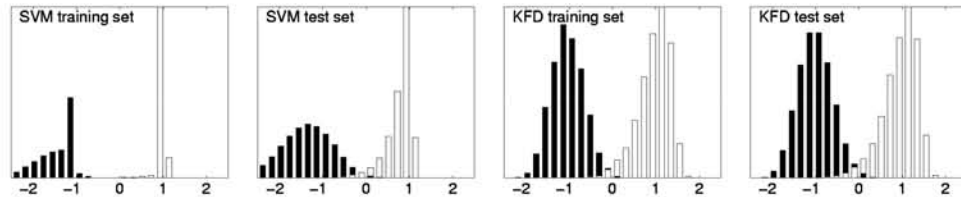

Figure 2: Comparison of output distributions on training and test set for SVM and KFD for optimal parameters on the ringnorm dataset (averaged over 100 different partitions). It is clearly observable, that the training and test set distributions for KFD are almost identical while they are considerable different for SVM.

**Performance**   To evaluate the performance of the various KFD approaches on real data sets we performed an extensive comparison to SVM[1]. The results in Table 2 show the

[1]Thanks to M. Zwitter and M. Soklic for the breast cancer data. All data sets used in the experiments can be obtained via `http://www.first.gmd.de/~raetsch/`.

average test error and the standard deviation of the averages' estimation, over 100 runs with different realizations of the datasets. To estimate the necessary parameters, we ran 5-fold cross validation on the first five realizations of the training sets and took the model parameters to be the median over the five estimates (see [7] for details of the experimental setup).

From Table 2 it can be seen that both, SVM and the KFD variants on average perform equally well. In terms of (4) KFD denotes the formulation with quadratic regularizer, SKFD with $l_1$–regularizer, and LSKFD with $l_1$–regularizer and $l_1$ loss on $\xi$. The comparable performance might be seen as an indicator, that maximizing the smallest margin or the average margin does not make a big difference on the data sets studied. The same seems to be true for using different regularizer and loss functions. Noteworthy is the significantly higher degree of sparsity for KFD.

**Regression**  Just to show that the proposed KFD regression works in principle, we conducted a toy experiment on the sinc function (cf. Figure 3). In terms of the number of support vectors we obtain similarly sparse results as with RVMs [11], i.e. a much smaller number of non–zero coefficients than in SVM regression. A thorough evaluation is currently being carried out.

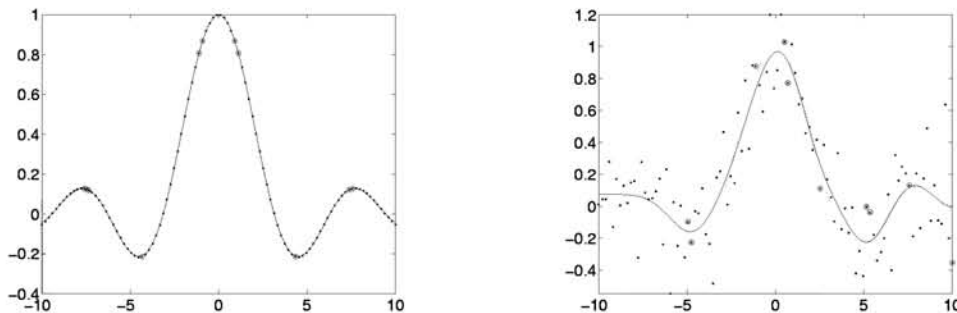

Figure 3: Illustration of KFD regression. The left panel shows a fit to the noise–free sinc function sampled on 100 equally spaced points, the right panel with Gaussian noise of std. dev. 0.2 added. In both cases we used RBF–kernel $\exp(-\|x-y\|^2/c)$ of width $c = 4.0$ and $c = 3.0$, respectively. The regularization was $C = 0.01$ and $C = 0.1$ (small dots training samples, circled dots SVs).

|          | SVM |  | KFD |  | SKFD |  | LSKFD |  |
|----------|-----|-----|-----|-----|------|------|-------|------|
| Banana   | 11.5±0.07 | (78%) | 10.8±0.05 |  | 11.2±0.48 | (86%) | **10.6±0.04** | (92%) |
| B.Cancer | 26.0±0.47 | (42%) | 25.8±0.46 |  | **25.2±0.44** | (88%) | 25.8±0.47 | (88%) |
| Diabetes | 23.5±0.17 | (57%) | 23.2±0.16 |  | **23.1±0.18** | (97%) | 23.6±0.18 | (97%) |
| German   | **23.6±0.21** | (58%) | 23.7±0.22 |  | **23.6±0.23** | (96%) | 24.1±0.23 | (98%) |
| Heart    | **16.0±0.33** | (51%) | 16.1±0.34 |  | 16.4±0.31 | (88%) | **16.0±0.36** | (96%) |
| Ringnorm | 1.7±0.01 | (62%) | **1.5±0.01** |  | 1.6±0.01 | (85%) | **1.5±0.01** | (94%) |
| F.Sonar  | **32.4±0.18** | (9%) | 33.2±0.17 |  | 33.4±0.17 | (67%) | 34.4±0.23 | (99%) |
| Thyroid  | 4.8±0.22 | (79%) | **4.2±0.21** |  | 4.3±0.18 | (88%) | 4.7±0.22 | (89%) |
| Titanic  | **22.4±0.10** | (10%) | 23.2±0.20 |  | 22.6±0.17 | (8%) | 22.5±0.20 | (95%) |
| Waveform | **9.9±0.04** | (60%) | **9.9±0.04** |  | 10.1±0.04 | (81%) | 10.2±0.04 | (96%) |

Table 2: Comparison between KFD, sparse KFD (SKFD), sparse KFD with linear loss on $\xi$ (LSKFD), and SVMs (see text). All experiments were carried out with RBF–kernels $\exp(-\|x-y\|^2/c)$. Best result in bold face, second best in italics. The numbers in brackets denote the fraction of expansions coefficients which were zero.

# 6 Conclusion and Outlook

In this work we showed how KFD can be reformulated as a mathematical programming problem. This allows a better understanding of KFD and interesting extensions: First, a probabilistic interpretation gives new insights about connections to RVM, SVM and regularization properties. Second, using a Laplacian prior, i.e. a $l_1$ regularizer yields the sparse algorithm SKFD. Third, the more general modeling permits a very natural KFD algorithm for regression. Finally, due to the quadratic programming formulation, we can use tricks known from SVM literature like chunking or active set methods for solving the optimization problem. However the optimal choice of a working set is not completely resolved and is still an issue of ongoing research. In this sense sparse KFD inherits some of the most appealing properties of both, SVM and RVM: a unique, mathematical programming solution from SVM and a higher sparsity together with interpretable outputs from RVM.

Our experimental studies show a competitive performance of our new KFD algorithms if compared to SVMs. This indicates that neither the margin nor sparsity nor a specific output distribution *alone* seem to be responsible for the good performance of kernel–machines. Further theoretical and experimental research is therefore needed to learn more about this interesting question. Our future research will also investigate the role of output distributions and their difference between training and test set.

**Acknowledgments** This work was partially supported by grants of the DFG (JA 379/7-1,9-1). Thanks to K. Tsuda for helpful comments and discussions.

# References

[1] G. Baudat and F. Anouar. Generalized discriminant analysis using a kernel approach. *Neural Computation*, 12(10):2385–2404, 2000.

[2] B.E. Boser, I.M. Guyon, and V.N. Vapnik. A training algorithm for optimal margin classifiers. In D. Haussler, editor, *Proceedings of the 5th Annual ACM Workshop on Computational Learning Theory*, pages 144–152, 1992.

[3] J.H. Friedman. Regularized discriminant analysis. *Journal of the American Statistical Association*, 84(405):165–175, 1989.

[4] S. Mika, G. Rätsch, J. Weston, B. Schölkopf, and K.-R. Müller. Fisher discriminant analysis with kernels. In Y.-H. Hu, J. Larsen, E. Wilson, and S. Douglas, editors, *Neural Networks for Signal Processing IX*, pages 41–48. IEEE, 1999.

[5] S. Mika, G. Rätsch, J. Weston, B. Schölkopf, A.J. Smola, and K.-R. Müller. Invariant feature extraction and classification in kernel spaces. In S.A. Solla, T.K. Leen, and K.-R. Müller, editors, *Advances in Neural Information Processing Systems 12*, pages 526–532. MIT Press, 2000.

[6] S. Mika, A.J. Smola, and B. Schölkopf. An improved training algorithm for kernel fisher discriminants. In *Proceedings AISTATS 2001*. Morgan Kaufmann, 2001. to appear.

[7] G. Rätsch, T. Onoda, and K.-R. Müller. Soft margins for AdaBoost. *Machine Learning*, 42(3):287–320, March 2001. also NeuroCOLT Technical Report NC-TR-1998-021.

[8] V. Roth and V. Steinhage. Nonlinear discriminant analysis using kernel functions. In S.A. Solla, T.K. Leen, and K.-R. Müller, editors, *Advances in Neural Information Processing Systems 12*, pages 568–574. MIT Press, 2000.

[9] B. Schölkopf, A.J. Smola, and K.-R. Müller. Nonlinear component analysis as a kernel eigenvalue problem. *Neural Computation*, 10:1299–1319, 1998.

[10] A. J. Smola. *Learning with Kernels*. PhD thesis, Technische Universität Berlin, 1998.

[11] M.E. Tipping. The relevance vector machine. In S.A. Solla, T.K. Leen, and K.-R. Müller, editors, *Advances in Neural Information Processing Systems 12*, pages 652–658. MIT Press, 2000.

[12] V.N. Vapnik. *The nature of statistical learning theory*. Springer Verlag, New York, 1995.
